# Visual Navigation in a Robot using Zig-Zag Behavior

**M. Anthony Lewis**

Beckman Institute
405 N. Mathews Avenue
University of Illinois
Urbana, Illinois 61801

## Abstract

We implement a model of obstacle avoidance in flying insects on a small, monocular robot. The result is a system that is capable of rapid navigation through a dense obstacle field. The key to the system is the use of zigzag behavior to articulate the body during movement. It is shown that this behavior compensates for a parallax blind spot surrounding the focus of expansion normally found in systems without parallax behavior. The system models the cooperation of several behaviors: halteres-ocular response (similar to VOR), optomotor response, and the parallax field computation and mapping to motor system. The resulting system is neurally plausible, very simple, and should be easily hosted on aVLSI hardware.

## 1 INTRODUCTION

Srinivasan and Zhang (1993) describe behavioral evidence for two distinct movement detecting systems in bee: (1) A direction selective pathway with low frequency response characteristics serving the optomotor response and (2) A non-direction selective movement system with higher frequency response serving functions of obstacle avoidance and the 'tunnel centering' response where the animal seeks a flight path along the centerline of a narrow corridor. Recently, this parallel movement detector view has received support from anatomical evidence in fly (Douglass and Strausfeld, 1996). We are concerned here with the implications of using non-direction selective movement detectors for tasks such as obstacle avoidance.

A reasonable model of a non-direction selective pathway would be that this pathway is computing the absolute value of the optic flow, i.e. $s = \|[\dot{x}, \dot{y}]\|$ where $\dot{x}, \dot{y}$ are the components of the optic flow field on the retina at the point $[x, y]$.

What is the effect of using the absolute value of the flow field and throwing away direction information? In section 2 we analyze the effect of a non-direction selective movement field. We understand from this analysis that rotational information, and the limited dynamic range of real sensors contaminates the non-direction selective field and

probably prevents the use of this technique in an area around the direction heading of the observer.

One technique to compensate for this 'parallax blind spot' is by periodically changing the direction of the observer. Such periodic movements are seen in insects as well as lower vertebrates and it is suggestive that these movements may compensate for this basic problem.

In Section 3, we describe a robotic implementation using a crude non-direction selective movement detector based on a rectified temporal derivative of luminosity. Each 'neuron' in the model retina issues a vote to control the motors of the robot. This system, though seemingly naively simple, compares favorably with other robotic implementations that rely on the optic flow or a function of the optic flow (divergence). These techniques typically require a large degree of spatial temporal averaging and seem computationally complex. In addition, our model agrees better with with the biological evidence.

Finally, the technique presented here is amenable to implementation in custom aVLSI or mixed aVLSI/dVLSI chips. Thus it should be possible to build a subminiature visually guided navigation system with several (one?) low-power simple custom chips.

## 2 ANALYSIS OF NON-DIRECTION SELECTIVE MOVEMENT DETECTION SYSTEM

Let us assume a perspective projection

$$\begin{bmatrix} x \\ y \end{bmatrix} = \frac{\lambda}{Z}\begin{bmatrix} X \\ Y \end{bmatrix} \tag{1}$$

where $\lambda$ is the focal length of the lens, $X, Y, Z$ is the position of a point in the environment, and $x, y$ is the projection of that point on the retinal plane. The velocity of the image of a moving point in the world can be found by differentiating (1) with respect to time:

$$\begin{bmatrix} \dot{x} \\ \dot{y} \end{bmatrix} = \frac{\lambda}{Z^2}\begin{bmatrix} Z\dot{X} - X\dot{Z} \\ Z\dot{Y} - Y\dot{Z} \end{bmatrix} \tag{2}$$

If we assume that objects in the environment are fixed in relation to one-and-other and that the observer is moving with relative translational velocity ${}^c v_e = \begin{bmatrix} v_x & v_y & v_z \end{bmatrix}^T$ and relative rotational velocity ${}^c\Omega_e = \begin{bmatrix} \omega_x & \omega_y & \omega_z \end{bmatrix}^T$ to the environment given in frame $c$, a point in the environment has relative velocity:

$$ {}^c\dot{P} = \begin{bmatrix} \dot{X} \\ \dot{Y} \\ \dot{Z} \end{bmatrix} = -\left({}^c\Omega_e \times {}^cP + {}^cv_e\right) \tag{3}$$

Now substituting in (2):

$$\begin{bmatrix} \dot{x} \\ \dot{y} \end{bmatrix} = \frac{\lambda}{Z}\begin{bmatrix} -1 & 0 & x \\ 0 & -1 & y \end{bmatrix}{}^cv_e + \lambda\begin{bmatrix} xy & -1-x^2 & y \\ 1+y^2 & -xy & -x \end{bmatrix}{}^c\Omega_e \tag{4}$$

and taking the absolute value of the optic flow:

$$\sqrt{s}^2 = \sqrt{\frac{\lambda^2 v_z^2}{Z^2}\left[\left(-\alpha + x - \frac{Z}{v_z}\left(xy\omega_x + \omega_y(x^2+1) + y\omega_z\right)\right)^2 + \left(-\beta + y + \frac{Z}{v_z}\left(-\omega_x(y^2+1) + xy\omega_y + x\omega_z\right)\right)^2\right]} \tag{5}$$

where we have made the substitution: $\begin{bmatrix} \dot{X}/Z & \dot{Y}/Z \end{bmatrix} \to \begin{bmatrix} \alpha & \beta \end{bmatrix}$ (that is, the heading direction).

We can see that the terms involving $\begin{bmatrix} \omega_x & \omega_y & \omega_z \end{bmatrix}$ cannot be separated from the $x, y$ terms. If we assume that $\begin{bmatrix} \omega_x & \omega_y & \omega_z \end{bmatrix} = 0$ then we can rearrange the equation as:

$$\Delta^{-1}(s) = \frac{|T_z|}{|Z|} = \frac{|s|}{\lambda\sqrt{\left[(x-\alpha)^2 + (y-\beta)^2\right]}}$$ (6)

in the case of $Z$ translation. If $|T_z| = 0$ then we have:

$$\Delta^{-1}(s) = \frac{1}{|Z|} = \frac{|s|}{\lambda\sqrt{T_x^2 + T_y^2}}$$ (7)

this corresponds to the case of pure lateral translations. Locusts (as well as some vertebrates) use *peering* or side to side movements to gauge distances before jumping.

We call the quantity in (6) inverse relative depth. Under the correct circumstances it is equivalent to the reciprocal of time to contact.

Equation (6) can be restated as: $\Delta^{-1}(s) = g_i s$ where $g$ is a gain factor that depends on the current direction heading and the position in the retina. This gain factor can be implemented neurally as a shunting inhibition, for example.

This has the following implications. If the observer is using a non-direction sensitive movement detector then (A) it must rotationally stabilize its eyes (B) it must dynamically alter the gain of this information in a pathway between the retinal input and motor output or it must always have a constant direction heading and use constant gain factors.

In real systems there is likely to be imperfection in rotational stabilization of the observer as well as sensors with limited dynamic range. To understand the effect of these, let us assume that there is a base-line noise level $\delta$ and we assume that this defines a minimum threshold substituting $s = \delta$, we can find a level curve for the minimum detectability of an object, i.e.:

$$\frac{\delta|Z|}{\lambda|T_z|} = \sqrt{\left[(u-\alpha)^2 + (v-\beta)^2\right]}$$ (8)

Thus, for constant depth and for $\delta$ independent of the spatial position on the retina, the level curve is a circle. The circle increases in radius with increasing distance, and noise, and decreases with increasing speed. The circle is centered around the direction heading.

The solution to the problem of a 'parallax blind spot' is to make periodic changes of direction. This can be accomplished in an open loop fashion or, perhaps, in an image driven fashion as suggested by Sobey (1994).

# 3 ROBOT MODEL

Figure 1a is a photograph of the robot model. The robot's base is a Khepera Robot. The Khepera is a small wheeled robot a little over 2" in diameter and uses differential drive motors. The robot has been fitted with a solid-state gyro attached to its body. This gyroscope senses angular velocities about the body axis and is aligned with the axis of the camera joint. A camera, capable of rotation about an axis perpendicular to the ground plane, is also attached. The camera has a field of view of about 90° and can swing of ±90°. The angle of the head rotation is sensed by a small potentiometer.

For convenience, each visual process is implemented on a separate Workstation (SGI Indy) as a heavyweight process. Interprocess communication is via PVM distributed computing library. Using a distributed processing model, behaviors can be dynamically added and deleted facilitating analysis and debugging.

## 3.1 ROBOT CONTROL SYSTEM

The control is divided into control modules as illustrated in Fig 2. At the top of the drawing we see a gaze stabilization pathway. This uses a gyro (imitating a halteres organ) for stabilization of rapid head movements. In addition, a visual pathway, using direction selective movement detector (DSMD) maps is used for slower optomotor response. Each of the six maps uses correlation type detectors (Borst and Egelhaaf, 1989). Each map is

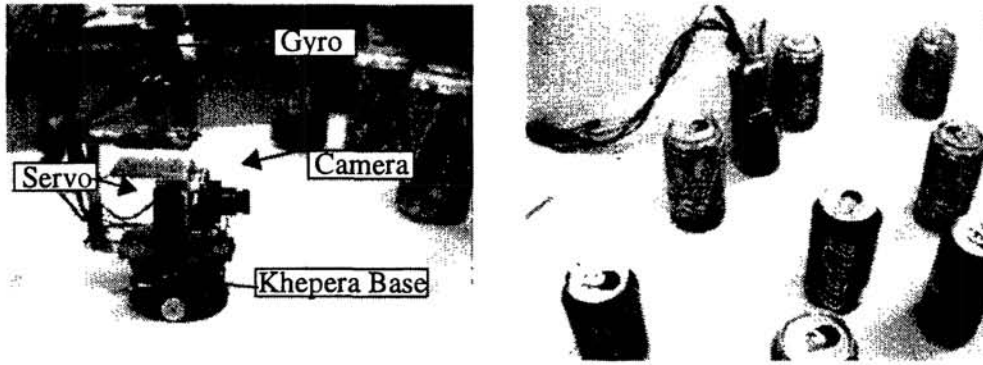

Figure 1. Physical setup. (A) Modified Khepera Robot with camera and gyro mounted. (B) Typical obstacle field run experiment.

tuned to a different horizontal velocity (three for left image translations and three for right image translations).

The lower half of the drawing shows the obstacle avoidance pathway. A crude non-direction selective movement detector is created using a simple temporal derivative. The use of this as a movement detector was motivated by the desire to eventually replace the camera front end with a Neuromorphic chip. Temporal derivative chips are readily available (Delbrück and Mead, 1991).

Next we reason that the temporal derivative gives a crude estimate of the absolute value of the optic flow. For example if we expect only horizontal flows then: $E_x \dot{x} = -E_t$ (Horn and Shunck, 1981). Here $E_t$ is the temporal derivative of the luminosity and $E_x$ is the spatial derivative. If we sample over a patch of the image, $E_x$ will take on a range of values. If we take the average rectified temporal derivative over a patch then $|\dot{x}| = \overline{|-E_t|}/\overline{|E_x|}$. Thus the average rectified temporal derivative over a patch will give a velocity proportional the absolute value of the optic flow.

In order to make the change to motor commands, we use a voting scheme. Each pixel in the nondirection selective movement detector field (NDSMD) votes on a direction for the robot. The left pixels for a right turn and the right pixels vote for a left turn. The left and right votes are summed. In certain experiments described below the difference of the left and right votes was used to drive the rotation of the robot. In others a symmetry breaking scheme was used. It was observed that with an object dead ahead of the robot, often the left and right activation would have high but nearly equal activation. In the symmetry breaking scheme, the side with the lower activation was further decrease by a factor of 50%. This admittedly ad hoc solution remarkably improved the performance *in the non-zig-zag case* as noted below.

The zig-zag behavior is implemented as a feedforward command to the motor system and is modeled as:

$$\omega_{ZigZag}^{Khepera} = \sin(\omega t)K$$

Finally, a constant forward bias is added to each wheel so the robot makes constant progress. $K$ is chosen empirically but in principle one should be able to derive it using the analysis in section 2.

As described above, the gaze stabilization module has control of head rotation and the zig-zag behavior and the depth from parallax behavior control the movement of the robot's body. During normal operation, the head may exceed the ±90° envelope defined by the mechanical system. This problem can be addressed in several ways among them are by making a body saccade to bring the body under the head or making a head saccade to align the head with the body. We choose the later approach solely because it seemed to work better in practice.

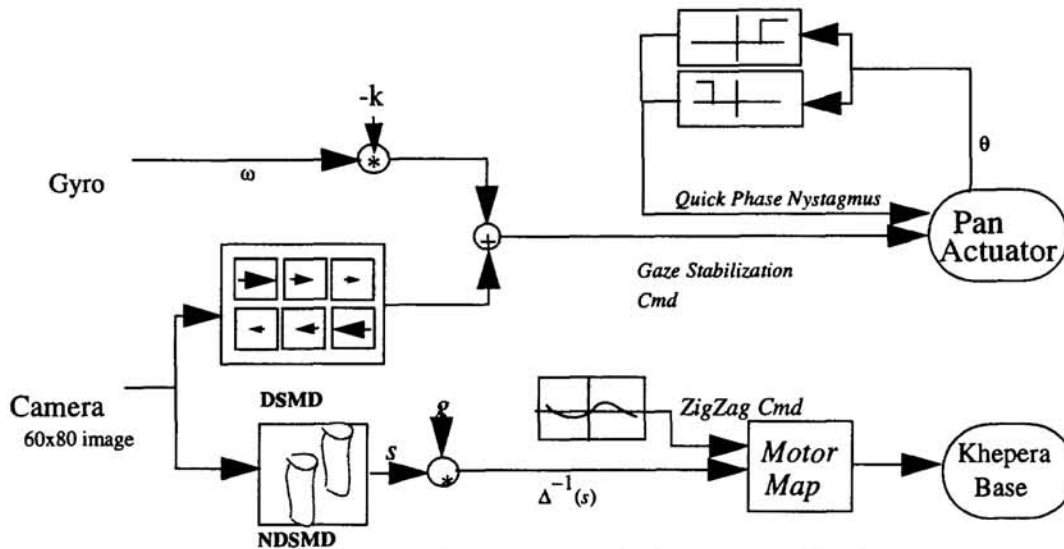

Figure 2. ZigZag Navigation model is composed of a gaze stabilization system (top) and an obstacle avoidance system (bottom). See text.

## 3.2 BIOLOGICAL INSPIRATION FOR MODEL

Course-grained visual pathways are modeled using inspiration from insect neurobiology. The model of depth from parallax is inspired by details given in Srinivasan & Zhang (1993) on work done in bees. Gaze stabilization using a fast channel, mediated by the halteres organs, and a slow optomotor response is inspired by a description of the blowfly *Calliphora* as reviewed by Hengstenberg (1991).

## 4 EXPERIMENTS

Four types of experimental setups were used. These are illustrated in Fig 3. In setup 1 the robot must avoid a dense field of obstacles (empty soda cans). This is designed to test the basic competence of this technique. In setup 2, thin dowels are place in the robot's path. This tests the spatial resolving capability of the robot. Likewise setup 3 uses a dense obstacle field with one opening replaced by a lightly textured surface.

Finally, experimental setup 4 uses a single small object (1cm black patch) and tests the distance at which the robot can 'lock-on' to a target. In this experiment, the avoidance field is sorted for a maximal element over a given threshold. A target cross is placed at this maximal element. The closest object should correspond with this maximal element. If a maximal element over a threshold is identified for a continuous 300ms and the target cross is on the correct target, the robot is stopped and its distance to the object is measured. The larger the distance, the better.

## 5 RESULTS

The results are described briefly here. In the setup 1 without the use of symmetry breaking, the scores were ZigZag: 10 Success, 0 Failures and the non-ZigZag: 4 Success and 6 Failures. With Symmetry Breaking installed the results were: ZigZag: 49 Success, 3 Failures and the non-ZigZag: 44 Success and 9 failures.

In the case palisades test: ZigZag: 22 Success, 4 Failures and the non-ZigZag: 14 Success and 11 failures.

In the false opening case: ZigZag: 8 Success, 2 Failures and the non-ZigZag: 6 Success and 4 Failures.

Finally, in the distance-to-lock setup, a lock was achieved at an average distance 21.3 CM (15 data points) for zigzag and 9.6 cm (15 data points) for the non-zigzag case.

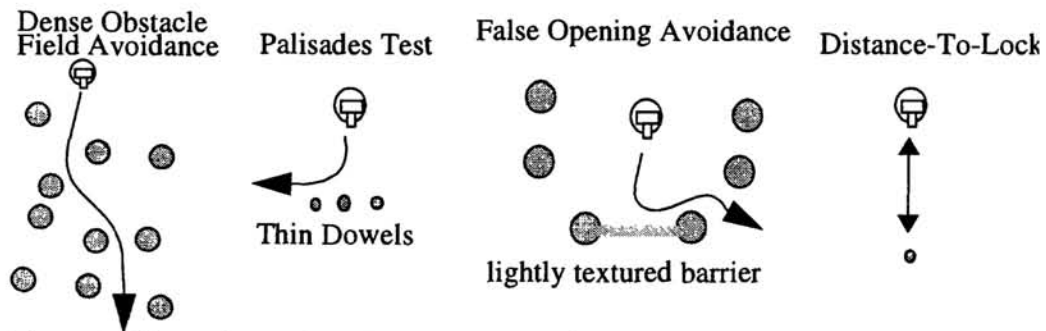

Figure 3. Illustrations of the four experimental setups.

We tentatively conclude that zig-zag behavior should improve performance in robot and in animal navigation.

## 6 DISCUSSION

In addition to the robotic implementation presented here, there have been many other techniques presented in the literature. Most relevant is Sobey (1994) who uses a zigzag behavior for obstacle avoidance. In this work, optic flow is computed through a process of discrete movements where 16 frames are collected, the robot stops, and the stored frames are analyzed for optic flow. The basic strategy is very clever: Always choose the next move in the direction of an identified object. The reasoning is that since we know the distance to the object in this direction, we can confidently move toward the object, stopping before collision. The basic idea of using zig-zag behavior is similar except that the zig-zagging is driven by perceptual input. In addition, the implementation requires estimation of the flow field requiring smoothing over numerous images. Finally, Sobey uses Optic Flow and we use the absolute value of the Optic Flow as suggested by biology.

Franceschini et al (1992) reports an analog implementation that uses elementary movement detectors. A unique feature is the non-uniform sampling and the use of three separate arrays. One array uses a sampling around the circumference. The other two sampling systems are mounted laterally on the robot and concentrate in the 'blind-spot' immediately in front of the robot. It is not clear that the strategy of using three sensor arrays, spatially separated, and direction selective movement detectors is in accord with the biological constraints.

Santos-Victor et al (1995) reports a system using optic flow and having lateral facing cameras. Here the authors were reproducing the centering reflex and did not focus on avoiding obstacles in front of the robot. Coombs and Roberts (1992,1993) use a similar technique. Weber et al (1996) describe wall following and stopping in front of an obstacle using an optic flow measure.

Finally, a number of papers report the use of flow field divergence, apparently first suggested by Nelson and Aloimonos (1989). This requires the computation of higher derivatives and requires significant smoothing. Even in this case, there is a problem of a 'parallax hole.' See Fig. 3 of that article, for example. In any case they did not implement their idea on a mobile robot. However, this approach has been followed up with an implementation in a robot by Camus et al (1996) reporting good results.

The system described here presents a physical model of insect like behavior integrated on a small robotic platform. Using results derived from an analysis of optic flow, we concluded that a zig-zag behavior in the robot would allow it to detect obstacles in front of the robot by periodically articulating the blind spot.

The complexity of the observed behavior and the simplicity of the control is striking. The robot is able to navigate through a field of obstacles, always searching out a freeway for movement.

The integrated behavior outlined here should be a good candidate for a neuromorphic implementation. A multichip (or single chip?) system could be envisioned using a relatively simple non-directional 2-d movement detector. Two arrays of perpendicular 1-d array of movement detectors should be sufficient for the optomotor response. This information could then be mapped to a circuit comprised of a few op-amp adder circuits and then sent to the head and body motors. Even the halteres organ could be simulated with a silicon fabricated gyroscope. The results would be an extremely compact robot capable of autonomous, visually guided navigation.

Finally, from our analysis of optic flow, we can make a reasonable prediction about the neural wiring in flying insects. The estimated depth of objects in the environment depends on where the object falls on the optic array as well as the ratio of translation to forward movement. Thus a bee or a fly should probably modulate its incoming visual signal to account for this time varying interpretation of the scene. We would predict that there should be motor information, related to the ratio of forward to lateral velocities would be projected to the non-directional selective motion detector array. This would allow a valid time varying interpretation of the scene in a zig-zagging animal.

### Acknowledgments

The author acknowledges the useful critique of this work by Narendra Ahuja, Mark Nelson, John Hart and Lucia Simo. Special thanks to Garrick Kremesic and Barry Stout who assisted with the experimental setup and the modification of the Khepera. The author acknowledges the support of ONR grant N000149610657. The author also acknowledges the loan of the Khepera from UCLA (NSF grant CDA-9303148).

### References

A. Borst and M. Egelhaaf (1989), Principles of Visual Motion Detection, Trends in Neurosciences, **12**(8):297-306

T. Camus, D. Coombs, M. Herman, and T.-H. Hong (1996), "Real-time Single-Workstation Obstacle Avoidance Using Only Wide-Field Flow Divergence", Proceedings of the 13th International Conference on Pattern Recognition. pp. 323-30 vol.3

D. Coombs and K. Roberts (1992), "'Bee-Bot': Using Peripheral Optical Flow to Avoid Obstacles", SPIE Vol 1825, Intelligent Robots and Computer Vision XI, pp 714-721.

D. Coombs and K. Roberts (1993), "Centering behavior using peripheral vision", Proc. 1993 IEEE Computer Society Conf. CVPR pp. 440-5, 16 refs. 1993

T. Delbrück and C. A. Mead (1991), Time-derivative adaptive silicon photoreceptor array. Proc. SPIE - Int. Soc. Opt. Eng. (USA). vol 1541, pp. 92-9.

J. K. Douglass and N. J. Strausfeld (1996), Visual Motion-Detection Circuits in Flies: Parallel Direction- and Non-Direction-Sensitive Pathways between the Medulla and Lobula Plate, J. of Neuroscience **16**(15):4551-4562.

N. Franceschini, J. M. Pichon and C. Blanes (1992), "From Insect Vision to Robot Vision", Phil. Trans. R. Soc Lond. B. **337**, pp 283-294.

R. Hengstenberg (1991), Gaze Control in the Blowfly *Calliphora*: a Multisensory, Two-Stage Integration Process, Seminars, in the Neurosciences, Vol3,pp 19-29.

B. K. P. Horn and B. G. Shunck (1981), "Determining Optic Flow", Artificial Intelligence, **17**(1-3):185-204.

R. C. Nelson and J. Y. Aloimonos (1989) Obstacle Avoidance Using Flow Field Divergence, IEEE Trans. on Pattern Anal. and Mach. Intel. **11**(10):1102-1106.

J. Santos-Victor, G. Sandini, F. Curotto and S. Garibaldi (1995), "Divergent Stereo in Autonomous Navigation: From Bees to Robots," Int. J. of Comp. Vis. **14**, pp 159-177.

P. J. Sobey (1994), "Active Navigation With a Monocular Robot" Biol. Cybern, **71**:433-440

M. V. Srinivasan and S. W. Zhang (1993), Evidence for Two Distinct Movement-Detecting Mechanisms in Insect Vision, Naturwissenschaften, **80**, pp 38-41.

K. Weber, S. Venkatash and M.V. Srinivasan (1996), "Insect Inspired Behaviours for the Autonomous Control of Mobile Robots" Proc. of ICPR'96, pp 156-160.